# An Analysis of Inference with the Universum

**Fabian H. Sinz**
Max Planck Institute for biological Cybernetics
Spemannstrasse 41, 72076, Tübingen, Germany
`fabee@tuebingen.mpg.de`

**Olivier Chapelle**
Yahoo! Research
Santa Clara, California
`chap@yahoo-inc.com`

**Alekh Agarwal**
University of California Berkeley
387 Soda Hall Berkeley, CA 94720-1776
`alekh@eecs.berkeley.edu`

**Bernhard Schölkopf**
Max Planck Institute for biological Cybernetics
Spemannstrasse 38, 72076, Tübingen, Germany
`bs@tuebingen.mpg.de`

## Abstract

We study a pattern classification algorithm which has recently been proposed by Vapnik and coworkers. It builds on a new inductive principle which assumes that in addition to positive and negative data, a third class of data is available, termed the *Universum*. We assay the behavior of the algorithm by establishing links with Fisher discriminant analysis and oriented PCA, as well as with an SVM in a projected subspace (or, equivalently, with a data-dependent reduced kernel). We also provide experimental results.

## 1  Introduction

Learning algorithms need to make assumptions about the problem domain in order to generalise well. These assumptions are usually encoded in the regulariser or the prior. A generic learning algorithm usually makes rather weak assumptions about the regularities underlying the data. An example of this is smoothness. More elaborate prior knowledge, often needed for a good performance, can be hard to encode in a regulariser or a prior that is computationally efficient too.

Interesting hybrids between both extremes are regularisers that depend on an additional set of data available to the learning algorithm. A prominent example of data-dependent regularisation is semi-supervised learning [1], where an additional set of unlabelled data, assumed to follow the same distribution as the training inputs, is tied to the regulariser using the so-called *cluster assumption*.

A novel form of data-dependent regularisation was recently proposed by [11]. The additional dataset for this approach is explicitly *not* from the same distribution as the labelled data, but represents a third — *neither* — class. This kind of dataset was first proposed by Vapnik [10] under the name *Universum*, owing its name to the intuition that the Universum captures a general backdrop against which a problem at hand is solved. According to Vapnik, a suitable set for this purpose can be thought of as a set of examples that belong to the same problem framework, but about which the resulting decision function should not make a strong statement.

Although initially proposed for transductive inference, the authors of [11] proposed an inductive classifier where the decision surface is chosen such that the Universum examples are located close to it. Implementing this idea into an SVM, different choices of Universa proved to be helpful in various classification tasks. Although the authors showed that different choices of Universa and loss functions lead to certain known regularisers as special cases of their implementation, there are still a few unanswered questions. On the one hand it is not clear whether the good performance of their algorithm is due to the underlying original idea, or just a consequence of the employed algorithmic

relaxation. On the other hand, except in special cases, the influence of the Universum data on the resulting decision hyperplane and therefore criteria for a good choice of a Universum is not known.

In the present paper we would like to address the second question by analysing the influence of the Universum data on the resulting function in the implementation of [11] as well as in a least squares version of it which we derive in section 2. Clarifying the regularising influence of the Universum on the solution of the SVM can give valuable insight into which set of data points might be a helpful Universum and how to obtain it.

The paper is structured as follows. After briefly deriving the algorithms in section 2 we show in section 3 that the algorithm of [11] pushes the normal of the hyperplane into the orthogonal complement of the subspace spanned by the principal directions of the Universum set. Furthermore, we demonstrate that the least squares version of the Universum algorithm is equivalent to a hybrid between kernel Fisher Discriminant Analysis and kernel Oriented Principal Component Analysis. In section 4, we validate our analysis on toy experiments and give an example how to use the geometric and algorithmic intuition gained from the analysis to construct a Universum set for a real world problem.

## 2 The Universum Algorithms

### 2.1 The Hinge Loss $\mathfrak{U}$-SVM

We start with a brief review of the implementation proposed in [11]. Let $\mathfrak{L} = \{(\mathbf{x}_1, y_1), ..., (\mathbf{x}_m, y_m)\}$ be the set of labelled examples and let $\mathfrak{U} = \{\mathbf{z}_1, ..., \mathbf{z}_q\}$ denote the set of Universum examples. Using the hinge loss $H_a[t] = \max\{0, a - t\}$ and $f_{\mathbf{w},b}(\mathbf{x}) = \langle \mathbf{w}, \mathbf{x} \rangle + b$, a standard SVM can compactly be formulated as

$$\min_{\mathbf{w},b} \quad \frac{1}{2}||\mathbf{w}||^2 + C_{\mathfrak{L}} \sum_{i=1}^{m} H_1[y_i f_{\mathbf{w},b}(\mathbf{x}_i)].$$

In the implementation of [11] the goal of bringing the Universum examples close to the separating hyperplane is realised by also minimising the cumulative $\varepsilon$-insensitive loss $I_\varepsilon[t] = \max\{0, |t| - \varepsilon\}$ on the Universum points

$$\min_{\mathbf{w},b} \quad \frac{1}{2}||\mathbf{w}||^2 + C_{\mathfrak{L}} \sum_{i=1}^{m} H_1[y_i f_{\mathbf{w},b}(\mathbf{x})] + C_{\mathfrak{U}} \sum_{j=1}^{q} I_\varepsilon[\, |f_{\mathbf{w},b}(\mathbf{z}_j)|\, ]. \tag{1}$$

Noting that $I_\varepsilon[t] = H_{-\varepsilon}[t] + H_{-\varepsilon}[-t]$, one can use the simple trick of adding the Universum examples twice with opposite labels and obtain an SVM like formulation which can be solved with a standard SVM optimiser.

### 2.2 The Least Squares $\mathfrak{U}$-SVM

The derivation of the least squares $\mathfrak{U}$-SVM starts with the same general regularised error minimisation problem

$$\min_{\mathbf{w},b} \quad \frac{1}{2}||\mathbf{w}||^2 + \frac{C_{\mathfrak{L}}}{2} \sum_{i=1}^{m} Q_{y_i}[f_{\mathbf{w},b}(\mathbf{x})] + \frac{C_{\mathfrak{U}}}{2} \sum_{j=1}^{q} Q_0[f_{\mathbf{w},b}(\mathbf{z}_j)]. \tag{2}$$

Instead of using the hinge loss, we employ the quadratic loss $Q_a[t] = ||t - a||_2^2$ which is used in the least squares versions of SVMs [9]. Expanding (2) in terms of slack variables $\boldsymbol{\xi}$ and $\boldsymbol{\vartheta}$ yields

$$\min_{\mathbf{w},b} \quad \frac{1}{2}||\mathbf{w}||^2 + \frac{C_{\mathfrak{L}}}{2} \sum_{i=1}^{m} \xi_i^2 + \frac{C_{\mathfrak{U}}}{2} \sum_{j=1}^{q} \vartheta_j^2 \tag{3}$$

$$\text{s.t.} \quad \langle \mathbf{w}, \mathbf{x}_i \rangle + b = y_i - \xi_i \text{ for } i = 1, ..., m$$

$$\langle \mathbf{w}, \mathbf{z}_j \rangle + b = 0 - \vartheta_j \text{ for } j = 1, ..., q.$$

Minimising the Lagrangian of (3) with respect to the primal variables $\mathbf{w}, b, \boldsymbol{\xi}$ and $\boldsymbol{\vartheta}$, and substituting their optimal values back into (3) yields a dual maximisation problem in terms of the Lagrange

multipliers $\boldsymbol{\alpha}$. Since this dual problem is still convex, we can set its derivative to zero and thereby obtain the following linear system

$$\left( \begin{array}{cc} 0 & \mathbf{1}^\top \\ \mathbf{1} & \mathbf{K} + \mathbf{C} \end{array} \right) \left( \begin{array}{c} b \\ \boldsymbol{\alpha} \end{array} \right) = \left( \begin{array}{c} 0 \\ \mathbf{y} \\ \mathbf{0} \end{array} \right),$$

Here, $\mathbf{K} = \left( \begin{array}{cc} \mathbf{K}_{\mathfrak{L},\mathfrak{L}} & \mathbf{K}_{\mathfrak{L},\mathfrak{U}} \\ \mathbf{K}_{\mathfrak{L},\mathfrak{U}}^\top & \mathbf{K}_{\mathfrak{U},\mathfrak{U}} \end{array} \right)$ denotes the kernel matrix between the input points in the sets $\mathfrak{L}$ and $\mathfrak{U}$, and $\mathbf{C} = \left( \begin{array}{cc} \frac{1}{C_{\mathfrak{L}}}\mathbf{I} & \mathbf{0} \\ \mathbf{0} & \frac{1}{C_{\mathfrak{U}}}\mathbf{I} \end{array} \right)$ an identity matrix of appropriate size scaled with $\frac{1}{C_{\mathfrak{L}}}$ in dimensions associated with labelled examples and $\frac{1}{C_{\mathfrak{U}}}$ for dimensions corresponding to Universum examples.

The solution $(\boldsymbol{\alpha}, b)$ can then be obtained by a simple matrix inversion. In the remaining part of this paper we denote the least squares SVM by $\mathfrak{U}_{\mathfrak{ls}}$-SVM.

### 2.3 Related Ideas

Although [11] proposed the first algorithm that explicitly refers to Vapnik's Universum idea, there exist related approaches that we shall mention briefly. The authors of [12] describe an algorithm for the one-vs-one strategy in multiclass learning that additionally minimises the distance of the separating hyperplane to the examples that are in neither of the classes. Although this is algorithmically equivalent to the $\mathfrak{U}$-SVM formulation above, their motivation is merely to sharpen the contrast between the different binary classifiers. In particular, they do not consider using a Universum for binary classification problems.

There are also two Bayesian algorithms that refer to *non-examples* or *neither class* in the binary classification setting. [8] gives a probabilistic interpretation for a standard hinge loss SVM by establishing the connection between the MAP estimate of a Gaussian process with a Gaussian prior using a covariance function $k$ and a hinge loss based noise model. In order to deal with the problem that the proposed likelihood does not integrate to one the author introduces a third — *the neither*— class, A similar idea is used by [4], introducing a third class to tackle the problem that unlabelled examples used in semi-supervised learning do not contribute to discriminative models $\mathbf{P}_{\mathsf{Y}|\mathsf{X}}(y_i|x_i)$ since the parameters of the label distribution are independent of input points with unknown, i.e., marginalised value of the label. To circumvent this problem, the authors of [4] introduce an additional — *neither* — class to introduce a stochastic dependence between the parameter and the unobserved label in the discriminative model. However, neither of the Bayesian approaches actually assigns an observed example to the introduced third class.

## 3 Analysis of the Algorithm

The following two sections analyse the geometrical relation of the decision hyperplane learnt with one of the Universum SVMs to the Universum set. It will turn out that in both cases the optimal solutions tend to make the normal vector orthogonal to the principal directions of the Universum. The extreme case where $\mathbf{w}$ is completely orthogonal to $\mathfrak{U}$, makes the decision function defined by $\mathbf{w}$ invariant to transformations that act on the subspace spanned by the elements of $\mathfrak{U}$. Therefore, the Universum should contain directions the resulting function should be invariant against.

In order to increase the readability we state all results for the linear case. However, our results generalise to the case where the $\mathbf{x}_i$ and $\mathbf{z}_j$ live in an RKHS spanned by some kernel.

### 3.1 $\mathfrak{U}$-SVM and Projection Kernel

For this section we start by considering a $\mathfrak{U}$-SVM with hard margin on the elements of $\mathfrak{U}$. Furthermore, we use $\varepsilon = 0$ for the $\varepsilon$-insensitive loss. After showing the equivalence to using a standard SVM trained on the orthogonal complement of the subspace spanned by the $\mathbf{z}_j$, we extend the result to the cases with soft margin on $\mathfrak{U}$.

**Lemma**  A $\mathfrak{U}$-SVM with $C_{\mathfrak{U}} = \infty, \varepsilon = 0$ is equivalent to training a standard SVM with the training points projected onto the orthogonal complement of $\text{span}\{\mathbf{z}_j - \mathbf{z}_0, \ \mathbf{z}_j \in \mathfrak{U}\}$, where $\mathbf{z}_0$ is an arbitrary element of $\mathfrak{U}$.

*Proof:* Since $C_{\mathfrak{U}} = \infty$ and $\varepsilon = 0$, any $\mathbf{w}$ yielding a finite value of (1) must fulfil $\langle \mathbf{w}, \mathbf{z}_j \rangle + b = 0$ for all $j = 1, ..., q$. So $\langle \mathbf{w}, \mathbf{z}_j - \mathbf{z}_0 \rangle = 0$ and $\mathbf{w}$ is orthogonal to $\text{span}\{\mathbf{z}_j - \mathbf{z}_0, \ \mathbf{z}_j \in \mathfrak{U}\}$. Let $P_{\mathfrak{U}\perp}$ denote the projection operator onto the orthogonal complement of that set. From the previous argument, we can replace $\langle \mathbf{w}, \mathbf{x}_i \rangle$ by $\langle P_{\mathfrak{U}\perp} \mathbf{w}, \mathbf{x}_i \rangle$ in the solution of (1) without changing it. Indeed, the optimal $\mathbf{w}$ in (1) will satisfy $\mathbf{w} = P_{\mathfrak{U}\perp} \mathbf{w}$. Since $P_{\mathfrak{U}\perp}$ is an orthogonal projection we have that $P_{\mathfrak{U}\perp} = P_{\mathfrak{U}\perp}^{\top}$ and hence $\langle P_{\mathfrak{U}\perp} \mathbf{w}, \mathbf{x}_i \rangle = \langle \mathbf{w}, P_{\mathfrak{U}\perp}^{\top} \mathbf{x}_i \rangle = \langle \mathbf{w}, P_{\mathfrak{U}\perp} \mathbf{x}_i \rangle$. Therefore, the optimisation problem in (1) is the same as a standard SVM where the $\mathbf{x}_i$ have been replaced by $P_{\mathfrak{U}\perp} \mathbf{x}_i$. $\qquad\square$

The special case the lemma refers to, clarifies the role of the Universum in the $\mathfrak{U}$-SVM. Since the resulting $\mathbf{w}$ is orthogonal to an affine space spanned by the Universum points, it is invariant against features implicitly specified by directions of large variance in that affine space. Picturing the $\langle \cdot, \mathbf{z}_j \rangle$ as filters that extract certain features from given labelled or test examples $\mathbf{x}$, using the Universum algorithms means suppressing the features specified by the $\mathbf{z}_j$.

Finally, we generalise the result of the lemma by dropping the hard constraint assumption on the Universum examples, i.e. we consider the case $C_{\mathfrak{U}} < \infty$. Let $\mathbf{w}^*$ and $b^*$ the optimal solution of (1). We have that

$$C_{\mathfrak{U}} \sum_{j=1}^{q} |\langle \mathbf{w}^*, \mathbf{z}_j \rangle + b^*| \geq C_{\mathfrak{U}} \min_b \sum_{j=1}^{q} |\langle \mathbf{w}^*, \mathbf{z}_j \rangle + b|.$$

The right hand side can be interpreted as an "$L_1$ variance". So the algorithm tries to find a direction $\mathbf{w}^*$ such that the variance of the projection of the Universum points on that direction is small. As $C_{\mathfrak{U}}$ approaches infinity this variance approaches 0 and we recover the result of the above lemma.

### 3.2 $\mathfrak{U}_{\mathfrak{ls}}$-SVM, Fisher Discriminant Analysis and Principal Component Analysis

In this section we present the relation of the $\mathfrak{U}_{\mathfrak{ls}}$-SVM to two classic learning algorithms: (kernel) oriented Principal Component Analysis (koPCA) and (kernel) Fisher discriminant analysis (kFDA) [5]. As it will turn out, the $\mathfrak{U}_{\mathfrak{ls}}$-SVM is equivalent to a hybrid between both up to a linear equality constraint. Since koPCA and kFDA can both be written as maximisation of a Rayleigh Quotient we start with the Rayleigh quotient of the hybrid

$$\max_{\mathbf{w}} \ \frac{\mathbf{w}^{\top} \overbrace{(\mathbf{c}^+ - \mathbf{c}^-)(\mathbf{c}^+ - \mathbf{c}^-)^{\top}}^{\text{from FDA}} \mathbf{w}}{\mathbf{w}^{\top}(C_{\mathfrak{L}} \underbrace{\sum_{k=\pm} \sum_{i \in \mathcal{I}^k} (\mathbf{x_i} - \mathbf{c}^k)(\mathbf{x_i} - \mathbf{c}^k)^{\top}}_{\text{from FDA}} + C_{\mathfrak{U}} \underbrace{\sum_{j=1}^{q} (\mathbf{z}_j - \tilde{\mathbf{c}})(\mathbf{z}_j - \tilde{\mathbf{c}})^{\top}}_{\text{from oPCA}})\mathbf{w}}.$$

Here, $\mathbf{c}^{\pm}$ denote the class means of the labelled examples and $\tilde{\mathbf{c}} = \frac{1}{2}(\mathbf{c}^+ + \mathbf{c}^-)$ is the point between them. As indicated in the equation, the numerator is exactly the same as in kFDA, i.e. the inter-class variance, while the denominator is a linear combination of the denominators from kFDA and koPCA, i.e. the inner class variances from kFDA and the noise variance from koPCA.

As noted in [6] the numerator is just a rank one matrix. For optimising the quotient it can be fixed to an arbitrary value while the denominator is minimised. Since the denominator might not have full rank it needs to be regularised [6]. Choosing the regulariser to be $||\mathbf{w}||^2$, the problem can be rephrased as

$$\min_{\mathbf{w}} \quad ||\mathbf{w}||^2 + \mathbf{w}^{\top} \left( C_{\mathfrak{L}} \sum_{k=\pm} \sum_{i \in \mathcal{I}^k} (\mathbf{x_i} - \mathbf{c}^k)(\mathbf{x_i} - \mathbf{c}^k)^{\top} + C_{\mathfrak{U}} \sum_{j=1}^{q} (\mathbf{z}_j - \tilde{\mathbf{c}})(\mathbf{z}_j - \tilde{\mathbf{c}})^{\top} \right) \mathbf{w} \quad (4)$$
$$\text{s.t.} \qquad\qquad\qquad\qquad\qquad \mathbf{w}^{\top}(\mathbf{c}^+ - \mathbf{c}^-) = 2$$

As we will see below this problem can further be transformed into a quadratic program

$$\min_{\mathbf{w}, b} \quad ||\mathbf{w}||^2 + C_{\mathfrak{L}} ||\boldsymbol{\xi}||^2 + C_{\mathfrak{U}} ||\boldsymbol{\vartheta}||^2 \quad (5)$$
$$\text{s.t.} \quad \langle \mathbf{w}, \mathbf{x}_i \rangle + b = y_i + \xi_i \text{ for all } i = 1, ..., m$$
$$\langle \mathbf{w}, \mathbf{z}_j \rangle + b = \vartheta_j \text{ for all } j = 1, ..., q$$
$$\boldsymbol{\xi}^{\top} \mathbf{1}^k = 0 \text{ for } k = \pm.$$

Ignoring the constraint $\boldsymbol{\xi}^{\top} \mathbf{1}^k = 0$, this program is equivalent to the quadratic program (3) of the $\mathfrak{U}_{\mathfrak{ls}}$-SVM. The following lemma establishes the relation of the $\mathfrak{U}_{\mathfrak{ls}}$-SVM to kFDA and koPCA.

**Lemma** For given $C_{\mathfrak{L}}$ and $C_{\mathfrak{U}}$ the optimisation problems (4) and (5) are equivalent.

*Proof:* Let $\mathbf{w}$, $b$, $\boldsymbol{\xi}$ and $\boldsymbol{\vartheta}$ the optimal solution of (5). Combining the first and last constraint, we get $\mathbf{w}^\top \mathbf{c}^\pm + b \mp 1 = 0$. This gives us $\mathbf{w}^\top(\mathbf{c}^+ - \mathbf{c}^-) = 2$ as well as $b = -\mathbf{w}^\top \tilde{\mathbf{c}}$. Plugging $\boldsymbol{\xi}$ and $\boldsymbol{\vartheta}$ in (5) and using this value of $b$, we obtain the objective function (4). So we have proved that the minimum value of (4) is not larger than the one of (5).

Conversely, let $\mathbf{w}$ be the optimal solution of (4). Let us choose $b = -\mathbf{w}^\top \tilde{\mathbf{c}}$, $\xi_i = \mathbf{w}^\top \mathbf{x}_i + b - y_i$ and $\vartheta_j = \mathbf{w}^\top \mathbf{z}_j + b$. Again both objective functions are equal. We just have to check that $\sum_{i:\, y_i = \pm 1} \xi_i = 0$. But because $\mathbf{w}^\top(\mathbf{c}^+ - \mathbf{c}^-) = 2$, we have

$$\frac{1}{m_\pm} \sum_{i:\, y_i = \pm 1} \xi_i = \mathbf{w}^\top \mathbf{c}^\pm + b \mp 1 = \mathbf{w}^\top \mathbf{c}^\pm - \frac{\mathbf{w}^\top(\mathbf{c}^+ + \mathbf{c}^-)}{2} \mp 1 = \frac{\mathbf{w}^\top(\mathbf{c}^\pm - \mathbf{c}^\mp)}{2} \mp 1 = 0. \square$$

The above lemma establishes a relation of the $\mathfrak{U}_{\mathfrak{l}\mathfrak{s}}$-SVM to two classic learning algorithms. This further clarifies the role of the Universum set in the algorithmic implementation of Vapnik's idea as proposed by [11]. Since the noise covariance matrix of koPCA is given by the covariance of the Universum points centered on the average of the labelled class means, the role of the Universum as a data-dependent specification of principal directions of invariance is affirmed.

The koPCA term also shows that both the position and covariance structure are crucial to a good Universum. To see this, we rewrite $\sum_{j=1}^q (\mathbf{z}_j - \tilde{\mathbf{c}})(\mathbf{z}_j - \tilde{\mathbf{c}})^\top$ as $\sum_{j=1}^q (\mathbf{z}_j - \tilde{\mathbf{z}})(\mathbf{z}_j - \tilde{\mathbf{z}})^\top + q(\tilde{\mathbf{z}} - \tilde{\mathbf{c}})(\tilde{\mathbf{z}} - \tilde{\mathbf{c}})^\top$, where $\tilde{\mathbf{z}} = \frac{1}{q}\sum_{j=1}^q \mathbf{z}_j$ is the Universum mean. The additive relationship between covariance of Universum about its mean, and the distance between Universum and training sample means projected onto $\mathbf{w}$ shows that either quantity can dominate depending on the data at hand.

In the next section, we demonstrate the theoretical results of this section on toy problems and give an example how to use the insight gained from this section to construct an appropriate Universum.

## 4 Experiments

### 4.1 Toy Experiments

The theoretical results of section 3 show that the covariance structure of the Universum as well as its absolute position influence the result of the learning process. To validate this insight on toy data, we sample ten labelled sets of size $20, 50, 100$ and $500$ from two fifty-dimensional Gaussians. Both Gaussians have a diagonal covariance that has low standard deviation ($\sigma_{1,2} = 0.08$) in the first two dimensions and high standard deviation ($\sigma_{3,\ldots,50} = 10$) in the remaining 48. The two Gaussians are displaced such that the mean of $\mu_i^\pm = \pm 0.3$ exceeds the standard deviation by a factor of 3.75 in the first two dimensions but was 125 times smaller in the remaining ones. The values are chosen such that the Bayes risk is approx. $5\%$. Note, that by construction the first two dimensions are most discriminative.

We construct two kinds of Universa for this toy problem. For the first kind we use a mean zero Gaussian with the same covariance structure as the Gaussians for the labelled data ($\sigma_{3,\ldots,50} = 10$), but with varied degree of anisotropy in the first two dimensions ($\sigma_{1,2} = 0.1, 1.0, 10$). According to the results of section 3 the Universa should be more helpful for larger anisotropy. For the second kind of Universa we use the same covariance as the labelled classes but shifted them along the line between the means of the labelled Gaussians. This kind of Universa should have a positive effect on the accuracy for small displacements but that effect should vanish with increasing amount of translation.

Figure 1 shows the performance of a linear $\mathfrak{U}$-SVMs for different amounts of training and Universum data. In the top row, the degree of isotropy increases from left to right, whereas $\sigma = 10$ refers to the complete isotropic case. In the bottom row, the amount of translation increases from left to right. As expected, performance converges to the performance of an SVM for high isotropy $\sigma$ and large translations $t$. Note, that large translations do not affect the accuracy as much as a high isotropy. However, this might be due to the fact the variance along the principal components of the Universum is much larger in magnitude than the applied shift. We obtained similar results for the $\mathfrak{U}_{\mathfrak{l}\mathfrak{s}}$-SVM. Also, the effect remains when employing an RBF kernel.

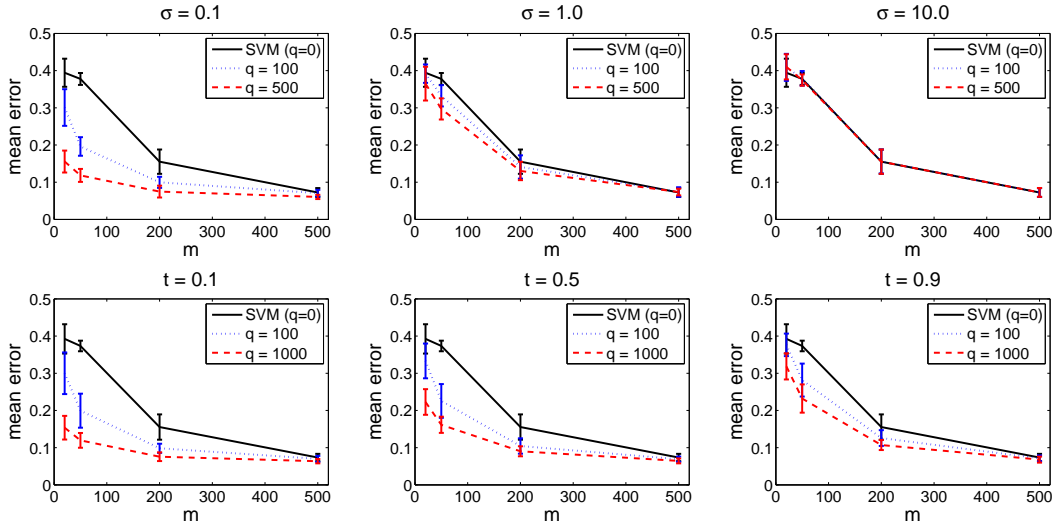

Figure 1: Learning curves of linear $\mathfrak{U}$-SVMs for different degrees of isotropy $\sigma$ and different amounts of translation $\mathbf{z} \mapsto \mathbf{z} + \frac{t}{2} \cdot (\mathbf{c}^+ - \mathbf{c}^-)$. With increasing isotropy and translation the performance of the $\mathfrak{U}$-SVMs converges to the performance of a normal SVM.

| Universum | 0 | 1 | 2 | 3 | 4 | 6 | 7 | 9 |
|---|---|---|---|---|---|---|---|---|
| Test error | 1.234 | 1.313 | 1.399 | 1.051 | 1.246 | 1.111 | 1.338 | 1.226 |
| Mean output | 0.406 | -0.708 | -0.539 | -0.031 | -0.256 | 0.063 | -0.165 | -0.360 |
| Angle | 81.99 | 85.57 | 79.49 | 69.74 | 79.75 | 81.02 | 82.72 | 77.98 |

Table 1: See text for details. Without Universum, test error is 1.419%. The correlation between the test error and the absolute value of the mean output (resp. angle) is 0.71 (resp 0.64); the $p$-value (i.e the probability of observing such a correlation by chance) is 3% (resp 5.5%). Note that for instance that digits 3 and 6 are the best Universum and they are also the closest to the decision boundary.

## 4.2 Results on MNIST

Following the experimental work from [11], we took up the task of distinguishing between the digits 5 and 8 on MNIST data. Training sets of size 1000 were used, and other digits served as Universum data. Using different digits as universa, we recorded the *test error* (in percentage) of $\mathfrak{U}$-SVM. We also computed the *mean output* (i.e. $\langle \mathbf{w}, \mathbf{x} \rangle + b$) of a normal SVM trained for binary classification between the digits 5 and 8, measured on the points from the Universum class. Another quantity of interest measured was the *angle between covariance matrices* of training and Universum data in the feature space. Note that for two covariance matrices $\mathbf{C}_X$ and $\mathbf{C}_Y$ corresponding to matrices $X$ and $Y$ (centered about their means), the cosine of the angle is defined as $\text{trace}(\mathbf{C}_X \mathbf{C}_Y)/\sqrt{\text{trace}(\mathbf{C}_X^2)\text{trace}(\mathbf{C}_Y^2)}$. This quantity can be computed in feature space as $\text{trace}(\mathbf{K}_{XY}\mathbf{K}_{XY}^\top)/\sqrt{\text{trace}(\mathbf{K}_{XX}^2)\text{trace}(\mathbf{K}_{YY}^2)}$, with $\mathbf{K}_{XY}$ the kernel matrix between the sets $X$ and $Y$. These quantities have been documented in Table 1. All the results reported are averaged over 10-folds of cross-validation, with $C = C_{\mathfrak{U}} = 100$, and $\varepsilon = 0.01$.

## 4.3 Classification of Imagined Movements in Brain Computer Interfaces

*Brain computer interfaces (BCI)* are devices that allow a user to control a computer by merely using his brain activity [3]. The user indicates different states to a computer system by deliberately changing his state of mind according to different experimental paradigms. These states are to be detected by a classifier. In our experiments, we used data from electroencephalographic recordings (EEG) with a imagined-movement paradigm. In this paradigm the patient imagines the movement of his left or right hand for indicating the respective state. In order to reverse the spatial blurring of the brain activity by the intermediate tissue of the skull, the signals from all sensors are demixed via

| | | DATA I | | |
|---|---|---|---|---|
| **Algorithm** | $\mathfrak{U}$ | **FS** | **JH** | **JL** |
| SVM | $\emptyset$ | $40.00 \pm 7.70$ | $40.00 \pm 11.32$ | $30.00 \pm 15.54$ |
| $\mathfrak{U}$-SVM | $\mathfrak{U}_{C3}$ | $41.33 \pm 7.06\ (0.63)$ | $34.58 \pm 9.22\ (0.07)$ | $30.56 \pm 17.22\ (1.00)$ |
| | $\mathfrak{U}_{nm}$ | $39.67 \pm 8.23\ (1.00)$ | $37.08 \pm 11.69\ (0.73)$ | $30.00 \pm 16.40\ (1.00)$ |
| LS-SVM | $\emptyset$ | $41.00 \pm 7.04$ | $40.42 \pm 11.96$ | $30.56 \pm 15.77$ |
| $\mathfrak{U}_{ls}$-SVM | $\mathfrak{U}_{C3}$ | $40.67 \pm 7.04\ (1.00)$ | $37.08 \pm 7.20\ (0.18)$ | $31.11 \pm 17.01\ (1.00)$ |
| | $\mathfrak{U}_{nm}$ | $40.67 \pm 6.81\ (1.00)$ | $37.92 \pm 12.65\ (1.00)$ | $30.00 \pm 15.54\ (1.00)$ |
| | | DATA II | | |
| | | **S1** | **S2** | **S3** |
| SVM | $\emptyset$ | $12.35 \pm 6.82$ | $35.29 \pm 13.30$ | $35.26 \pm 14.05$ |
| $\mathfrak{U}$-SVM | $\mathfrak{U}_{C3}$ | $13.53 \pm 6.83\ (0.63)$ | $32.94 \pm 11.83\ (0.63)$ | $35.26 \pm 14.05\ (1.00)$ |
| | $\mathfrak{U}_{nm}$ | $12.35 \pm 7.04\ (1.00)$ | $27.65 \pm 14.15\ (0.13)$ | $36.84 \pm 13.81\ (1.00)$ |
| LS-SVM | $\emptyset$ | $13.53 \pm 8.34$ | $33.53 \pm 13.60$ | $34.21 \pm 12.47$ |
| $\mathfrak{U}_{ls}$-SVM | $\mathfrak{U}_{C3}$ | $12.94 \pm 6.68\ (1.00)$ | $32.35 \pm 10.83\ (0.38)$ | $35.79 \pm 15.25\ (1.00)$ |
| | $\mathfrak{U}_{nm}$ | $16.47 \pm 7.74\ (0.50)$ | $31.18 \pm 13.02\ (0.69)$ | $35.79 \pm 15.25\ (1.00)$ |

Table 2: Mean zero-one test error scores for the BCI experiments. The mean was taken over ten single error scores. The $p$-value for a two-sided sign test against the SVM error scores are given in brackets.

an independent component analysis (ICA) applied to the concatenated lowpass filtered time series of all recording channels [2].

In the experiments below we used two BCI datasets. For the first set (DATA I) we recorded the EEG activity from three healthy subjects for an imagined movement paradigm as described by [3]. The second set (DATA II) contains EEG signals from a similar paradigm [7].

We constructed two kind of Universa. The first Universum, $\mathfrak{U}_{C3}$ consists of recordings from a third condition in the experiments that is not related to imagined movements. Since variations in signals from this condition should not carry any useful information about imagined movement task, the classifier should be invariant against them. The second Universum $\mathfrak{U}_{nm}$ is physiologically motivated. In the case of the imagined-movement paradigm the relevant signal is known to be in the so called $\alpha$-band from approximately $10 - 12$Hz and spatially located over the motor cortices. Unfortunately, signals in the $\alpha$-band are also related to visual activity and independent components can be found that have a strong influence from sensors over the visual cortex. However, since ICA is unsupervised, those independent components could still contain discriminative information. In order to make the learning algorithm prefer the signals from the motor cortex, we construct a Universum $\mathfrak{U}_{nm}$ by projecting the labelled data onto the independent components that have a strong influence from the visual cortex.

The machine learning experiments were carried out in two nested cross validation loops, where the inner loop was used for model selection and the outer for testing. We exclusively used a linear kernel. Table 2 shows the mean zero-one loss for DATA I and DATA II and the constructed Universa.

On the DATA I dataset, there is no improvement in the error rates for the subjects FS and JL compared to an SVM without Universum. Therefore, we must assume that the employed Universa did not provide helpful information in those cases. For subject JH, $\mathfrak{U}_{C3}$ and $\mathfrak{U}_{nm}$ yield an improvement for both Universum algorithms. However, the differences to the SVM error scores are not significantly better according to a two-sided sign test. The $\mathfrak{U}_{ls}$-SVM performs worse than the $\mathfrak{U}$-SVM in almost all cases.

On the DATA II dataset, there was an improvement only for subject S2 using the $\mathfrak{U}$-SVM with the $\mathfrak{U}_{nm}$ and $\mathfrak{U}_{C3}$ Universum (8% and 3% improvement respectively). However, also those differences are not significant. As already observed for the DATA I dataset, the $\mathfrak{U}_{ls}$-SVM performs constantly worse than its hinge loss counterpart.

The better performance of the $\mathfrak{U}_{nm}$ Universum on the subjects JH and S2 indicates that additional information about the usefulness of features might in fact help to increase the accuracy of the classifier. The regularisation constant $C_{\mathfrak{U}}$ for the Universum points was chosen $C = C_{\mathfrak{U}} = 0.1$ in both cases. This means that the non-orthogonality of $\mathbf{w}$ on the Universum points was only weakly

penalised, but had equal priority to classifying the labelled examples correctly. This could indicate that the spatial filtering by the ICA is not perfect and discriminative information might be spread over several independent components, even over those that are mainly non-discriminative. Using the $\mathfrak{U}_{nm}$ Universum and therefore gently penalising the use of these non-discriminative features can help to improve the classification accuracy, although the factual usefulness seems to vary with the subject.

## 5  Conclusion

In this paper we analysed two algorithms for inference with a Universum as proposed by Vapnik [10]. We demonstrated that the $\mathfrak{U}$-SVM as implemented in [11] is equivalent to searching for a hyperplane which has its normal lying in the orthogonal complement of the space spanned by Universum examples. We also showed that the corresponding least squares $\mathfrak{U}_{ls}$-SVM can be seen as a hybrid between the two well known learning algorithms kFDA and koPCA where the Universum points, centered between the means of the labelled classes, play the role of the noise covariance in koPCA. Ideally the covariance matrix of the Universum should thus contain some important invariant directions for the problem at hand.

The position of the Universum set plays also an important role and both our theoretical and experimental analysis show that the behaviour of the algorithm depends on the difference between the means of the labelled set and of the Universum set. The question of whether the main influence of the Universum comes from the position or the covariance does not have a clear answer and is probably problem dependent.

From a practical point, the main contribution of this paper is to suggest how to select a good Universum set: it should be such that it contains invariant directions and is positioned "in between" the two classes. Therefore, as can be partly seen from the BCI experiments, a good Universum dataset needs to be carefully chosen and cannot be an arbitrary backdrop as the name might suggest.

## References

[1] O. Chapelle, B. Schölkopf, and A. Zien, editors. *Semi-Supervised Learning*. MIT Press, Cambridge, MA, 2006.

[2] N. J. Hill, T. N. Lal, M. Schröder, T. Hinterberger, B. Wilhelm, F. Nijboer, U. Mochty, G. Widman, C. E. Elger, B. Schölkopf, A. Kübler, and N. Birbaumer. Classifying EEG and ECoG signals without subject training for fast bci implementation: Comparison of non-paralysed and completely paralysed subjects. *IEEE Transactions on Neural Systems and Rehabilitation Engineering*, 14(2):183–186, 06 2006.

[3] T. N. Lal. *Machine Learning Methods for Brain-Computer Interdaces*. PhD thesis, University Darmstadt, 09 2005. Logos Verlag Berlin MPI Series in Biological Cybernetics, Bd. 12 ISBN 3-8325-1048-6.

[4] Neil D. Lawrence and Michael I. Jordan. Gaussian processes and the null-category noise model. In A. Zien O. Chapelle, Bernhard Schölkopf, editor, *Semi-Supervised Learning*, chapter 8, pages 137–150. MIT University Press, 2006.

[5] S. Mika, G. Rätsch, J. Weston, B. Schölkopf, A. Smola, and K. Müller. Invariant feature extraction and classification in kernel spaces. In *Advances in Neural Information Processing Systems 12*, pages 526–532, 2000.

[6] Sebastian Mika, Gunnar Rätsch, and Klaus-Robert Müller. A mathematical programming approach to the kernel fisher algorithm. In *Advances in Neural Information Processing Systems, NIPS*, 2000.

[7] J. del R. Millán. On the need for on-line learning in brain-computer interfaces. IDIAP-RR 30, IDIAP, Martigny, Switzerland, 2003. Published in "Proc. of the Int. Joint Conf. on Neural Networks", 2004.

[8] P. Sollich. Probabilistic methods for support vector machines. In *Advances in Neural Information Processing Systems*, 1999.

[9] J. A. K. Suykens and J. Vandewalle. Least squares support vector machine classifiers. *Neural Processing Letters*, 9(3):293–300, 1999.

[10] V. Vapnik. Transductive Inference and Semi-Supervised Learning. In O. Chapelle, B. Schölkopf, and A. Zien, editors, *Semi-Supervised Learning*, chapter 24, pages 454–472. MIT press, 2006.

[11] J. Weston, R. Collobert, F. Sinz, L. Bottou, and V. Vapnik. Inference with the universum. In *Proceedings of the 23rd International Conference on Machine Learning*, page 127, 06/25/ 2006.

[12] P. Zhong and M. Fukushima. A new support vector algorithm. *Optimization Methods and Software*, 21:359–372, 2006.

